# Feature Selection Methods for Improving Protein Structure Prediction with Rosetta

**Ben Blum, Michael I. Jordan**
Department of Electrical Engineering and Computer Science
University of California at Berkeley
Berkeley, CA 94305
{bblum,jordan}@cs.berkeley.edu

**David E. Kim, Rhiju Das, Philip Bradley, David Baker**
Department of Genome Sciences
University of Washington
Seattle, WA 98195
{dekim, rhiju, pbradley, dabaker}@u.washington.edu

## Abstract

Rosetta is one of the leading algorithms for protein structure prediction today. It is a Monte Carlo energy minimization method requiring many random restarts to find structures with low energy. In this paper we present a *resampling* technique for structure prediction of small alpha/beta proteins using Rosetta. From an initial round of Rosetta sampling, we learn properties of the energy landscape that guide a subsequent round of sampling toward lower-energy structures. Rather than attempt to fit the full energy landscape, we use feature selection methods—both $L1$-regularized linear regression and decision trees—to identify structural features that give rise to low energy. We then enrich these structural features in the second sampling round. Results are presented across a benchmark set of nine small alpha/beta proteins demonstrating that our methods seldom impair, and frequently improve, Rosetta's performance.

## 1 Introduction

Protein structure prediction is one of the most important unsolved problems in biology today. With the wealth of genome data now available, it is of great interest to determine the structures of the proteins that genes encode. Proteins are composed of long chains of amino acid residues, of which there are twenty natural varieties. A gene encodes a specific amino acid sequence, which, when translated, folds into a unique three-dimensional conformation. The protein structure prediction problem is to predict this conformation (the protein's *tertiary structure*) from the amino acid sequence (the protein's *primary structure*). The biological function of a protein is dependent on its structure, so structure prediction is an important step towards function prediction. Potential applications of structure prediction range from elucidation of cellular processes to vaccine design.

Experimental methods for protein structure determination are costly and time-intensive, and the number of known protein sequences now far outstrips the capacity of experimentalists to determine their structures. Computational methods have been improving steadily and are approaching the level of resolution attainable in experiments. Structure prediction methods fall into two broad camps: comparative modeling, in which solved protein structures are known for one or more proteins with sequences similar to the target sequence ("homologs"), and ab initio modeling, in which no homologs are known. In this paper we concentrate on ab initio modeling, and specifically on the Rosetta algorithm [3].

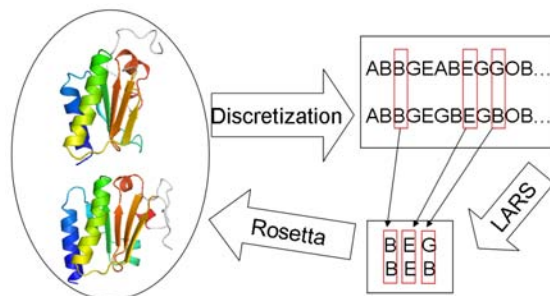

Figure 1: Flowchart of resampling method.

Rosetta is one of the leading methods for ab initio protein structure prediction today. Rosetta uses a Monte Carlo search procedure to minimize an energy function that is sufficiently accurate that the conformation found in nature (the "native" conformation) is generally the conformation with lowest Rosetta energy. Finding the global minimum of the energy function is very difficult because of the high dimensionality of the search space and the very large number of local minima. Rosetta employs a number of strategies to combat these issues, but the primary one is to perform a large number of random restarts. Thanks to a very large-scale distributed computing platform called Rosetta@home, composed of more than three hundred thousand volunteer computers around the world, up to several million local minima of the energy function ("decoys," in Rosetta parlance) can be computed for each target sequence.

Our work begins with the observation that a random-restart strategy throws away a great deal of information from previously computed local minima. In particular, previous samples from conformation space might suggest regions of uniformly lower energy; these are regions in which Rosetta may wish to concentrate further sampling. This observation is applicable to many global optimization problems, and past researchers have proposed a variety of methods for exploiting it, including fitting a smoothed *response surface* to the local minima already gathered [1] and learning to predict good starting points for optimization [2]. Unfortunately, conformation space is very high-dimensional and very irregular, so response surfaces do not generalize well beyond the span of the points to which they are fitted. Generally, the correct (or "native") structure will not be in the span of the points seen so far—if it were, the first round of Rosetta sampling would already have been successful.

We have developed an approach that sidesteps this limitation by explicitly recombining successful features of the models seen so far. No single local minimum computed in the first round of Rosetta search will have *all* the native features. However, many native features are present in at least *some* of the decoys. If these features can be identified and combined with each other, then sampling can be improved. Our approach has three steps, each mapping from one structural representation space to another (Figure 1). In the first step, we project the initial set of Rosetta models from continuous conformation space into a discrete feature space. The structural features that we have designed characterize significant aspects of protein structure and are largely sufficient to determine a unique conformation. In the second step, we use feature selection methods including both decision trees and Least Angle Regression (LARS) [4] to identify structural features that best account for energy variation in the initial set of models. We can then predict that certain of these features (generally, those associated with low energy) are present in the native conformation. In the third step, we use constrained Rosetta search to generate a set of models enriched for these key features.

## 2 Outline

In section 3, we outline a response surface approach and its shortcomings, and motivate the move to a feature-based representation. In section 4, we describe the features we use and the way that particular feature values are enforced in Rosetta search. This characterizes the way we map points from our discretized feature space back to continuous conformation space. In section 5, we describe the feature selection techniques we use to determine which features to enforce. In section 6, we

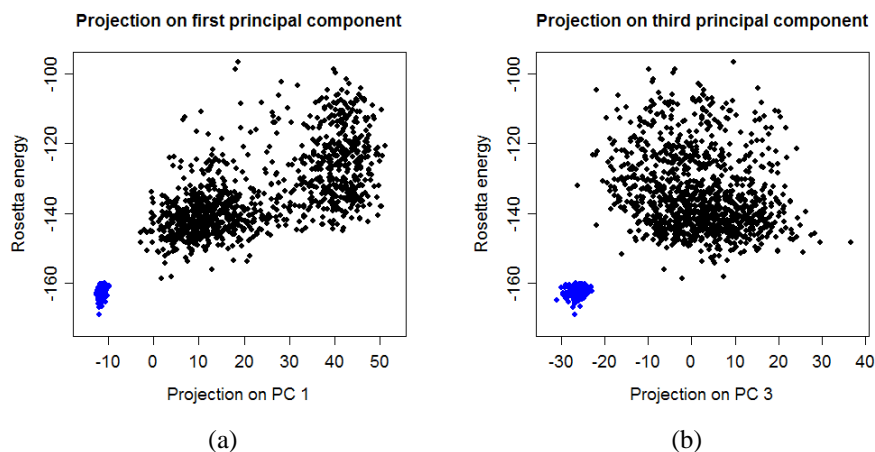

Figure 2: (a) Rosetta models (black) and relaxed natives (blue) projected onto the first principal component. (b) Models and natives projected onto the third principal component.

show the results of Rosetta search biased towards selected features. In section 7, we conclude with a discussion of the results achieved so far and of further work to be done.

## 3   Response Surface Methods

As an initial attempt at developing resampling methods for protein structure prediction, we investigated a response surface fitting approach. Our goal was to fit a smoothed energy surface to the Rosetta models seen so far and then to minimize this surface to find new starting points for local optimization of the Rosetta energy function.

The first task was to define the conformation space. The most natural space is defined in terms of the conformational degrees of freedom. Each residue in an amino acid sequence has two primary degrees of freedom: rotation around the $C_\alpha$–$N$ bond, referred to as the $\phi$ torsion angle, and rotation around the $C_\alpha$–$C$ bond, referred to as the $\psi$ torsion angle. However, it is difficult to fit a response surface in the space of torsion angles because the energy function is highly irregular in this space; a slight change in a single torsion angle typically causes large global structural changes, which in turn cause large energy changes. Instead, we took the three-dimensional coordinates of the backbone atoms as our conformation space, with all models in the set aligned to a reference model. There are four backbone atoms per residue and three coordinates per backbone atom, so an $n$-residue protein is represented by a $12n$-dimensional vector. Even for small proteins of only around 70 residues this space is very high-dimensional, but we found that most of the structural variation in sets of Rosetta models was captured by the first 10 principal components. Data were sufficient to fit a response surface in these 10 dimensions.

Along certain directions, energy gradients were detectable that pointed toward the native structure. One such direction was the first principal component for protein 1n0u (Figure 2.a; in this graph, the native structure is represented as an ensemble of Rosetta-minimized structures that started at the native conformation). However, in most directions the gradient did not point toward the natives (Figure 2.b). A response surface fitted to the Rosetta models shown in these graphs will therefore have high energy in the vicinity of the natives.

These observations suggest a new strategy: rather than fitting a response surface to all the dimensions jointly, we should identify a few dimensions that are associated with clear score gradients and make no claims about the other dimensions. This motivates a shift in philosophy: rather than predicting energy and minimizing, we wish to predict *features* of the native structure and then enforce them independently of each other.

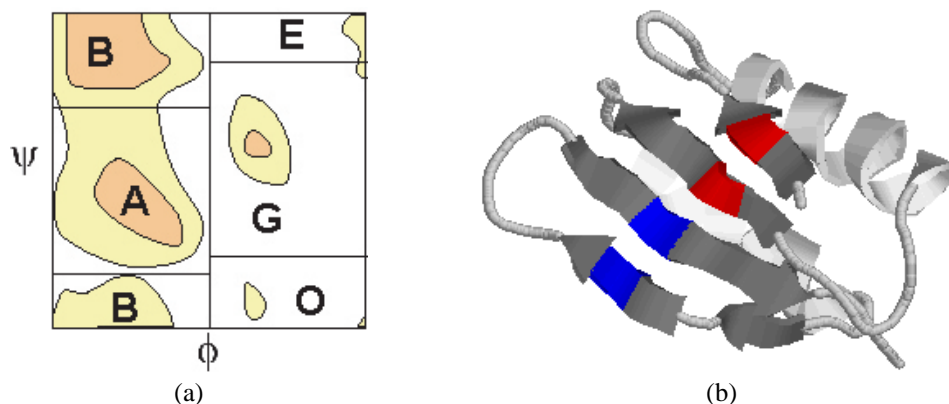

$$(a) \qquad\qquad\qquad\qquad\qquad (b)$$

Figure 3: (a) Bins in Ramachandran plot. (b) Structure of 1dcj. Two helices are visible behind a beta pleated sheat consisting of four strands, the bottommost three paired in the anti-parallel orientation and the topmost two paired in the parallel orientation. In this "cartoon" representation of structure, individual atoms are not rendered.

## 4  Structural features

For the purpose of the work described in this paper, we make use of two types of structural features: torsion angle features and beta contact features.

### 4.1  Torsion angle features

The observed values of the $\phi$ and $\psi$ angles for a single residue are strongly clustered in the database of solved protein structures (the *PDB*). Their empirical distribution is shown in a *Ramachandran plot*. In order to discretize the possible torsion angles for each residue, we divide the Ramachandran plot into five regions, referred to as "A," "B," "E," "G," and "O," (Figure 3.a). These regions are chosen to correspond roughly to clusters observed in the PDB. A protein with 70 amino acid residues has 70 torsion bin features, each with possible values A, B, E, G, and O.

The primary search move in Rosetta is a *fragment replacement* move: the conformation of a string of three or nine consecutive residues within the target sequence is replaced with the conformation of a similar subsequence from the PDB. A torsion angle feature can be constrained in Rosetta search by limiting the fragments to those which have torsion angles within the given bin at the given residue position. Strings of torsion features are referred to as *barcodes* in Rosetta, and the apparatus for defining and constraining them was developed in-house by Rosetta developers.

### 4.2  Beta contact features

Proteins exhibit two kinds of *secondary structure*, characterized by regular hydrogen bond patterns: alpha helices and beta pleated sheets (Figure 3.b). In alpha helices, the hydrogen bonds are all local, and are predicted fairly consistently by Rosetta. In beta sheets, however, the bonds can be between residues that are quite distant along the chain. A beta contact feature for residues $i$ and $j$ indicates the presence of two backbone hydrogen bonds between $i$ and $j$. We use the same definition of beta pairing as the standard secondary structure assignment algorithm DSSP [5]. The bonding pattern can be either parallel (as between the red residues in Figure 3.b) or antiparallel (as between the blue residues). Furthermore, the pleating can have one of two different orientations. A beta pairing feature is defined for every triple $(i, j, o)$ of residue numbers $i$ and $j$ and orientations $o \in$ {parallel, antiparallel}. The possible values of a beta pairing feature are X, indicating no pairing, and P1 or P2, indicating pleating of orientation 1 or 2, respectively.

Beta contact features are enforced in Rosetta by means of a technique called "jumping." A pseudo-backbone-bond is introduced between the two residues to be glued together. This introduces a closed loop into the backbone topology of the protein. Torsion angles within the loop can no longer be altered without breaking the loop, so, in order to permit further fragment replacements, a cut (or

"chainbreak") must be introduced somewhere else in the loop. The backbone now takes the form of a tree rather than a chain. After a Rosetta search trajectory terminates, an attempt is made to close the chainbreak with local search over several torsion angles on either side of it.

# 5 Prediction of native features

Let us transform our set of multi-valued features into a set of 0-1 valued features indicating whether or not a particular value for the feature is present. Let us assume that each binary feature $f$ has an independent energetic effect; if present, it brings with it an average energy bonus $b_f$. Under these assumptions, the full energy of a conformation $d$ is modelled as

$$E_0 + \sum_f d_f b_f + N,$$

where $E_0$ is a constant offset, $d_f$ is either 1 if the feature is present in $d$ or 0 if it is absent, and $N$ is Gaussian noise. This model is partially justified by the fact that the true energy is indeed a sum of energies from local interactions, and our features capture local structural information. Our hypothesis is that native features have lower energy on average even if other native features are not present.

In order to identify a small set of potentially native features, we use $L_1$ regularization, or lasso regression [6], to find a sparse model. The minimization performed is

$$\operatorname{argmin}_{(b,E_0)} \sum_{d \in \mathcal{D}} (E(d) - E_0 - \sum_f d_f b_f)^2 + C \sum_f |b_f|,$$

where $E(d)$ is the computed Rosetta energy of model $d$ and C is a regularization constant. The small set of features that receive non-zero weights are those that best account for energy variations in the population of decoys. These are the features we can most confidently predict to be native. The Least Angle Regression algorithm [4] allows us to efficiently compute the trajectory of solutions for all values of $C$ simultaneously. Experience with Rosetta has shown that constraining more than ten or fifteen torsion features can hamper search more than it helps; if there are very few fragments available for a given position that satisfy all torsion constraints, the lack of mobility at that position can be harmful. We typically take the point in the LARS trajectory that gives fifteen feature values.

## 5.1 Feature enforcement strategy

LARS gives us a set of feature values that have a strong effect on energy. Our hypothesis is that features strongly associated with lower energies—namely, those selected by LARS and given negative weights—are more likely to be native, and that features given positive weights by LARS are more likely to be non-native. This hypothesis is born out by our experiments on a benchmark set of 9 small alpha/beta proteins. The LARS prediction accuracy is given in Figure 4.a. This chart shows, for each protein, the fraction of LARS-selected features correctly labeled as native or non-native by the sign of the LARS weight. Fifteen LARS features were requested per protein. The more accurate "low energy leaf" predictions will be discussed in the next section.

It is clear from Figure 4.a that LARS is informative about native features for most proteins. However, we cannot rely wholly on its predictions. If we were simply to constrain every LARS feature, then Rosetta would never find the correct structure, since some incorrect features would be present in every model. Our resampling strategy is therefore to flip a coin at the beginning of the Rosetta run to decide whether or not to constrain a particular LARS feature. Coins are flipped independently for each LARS feature. Resampling improves on unbiased Rosetta sampling if the number of viable runs (runs in which no non-native features are enforced) is sufficiently high that the benefits from the enforcement of native features are visible. We have achieved some success by enforcing LARS features with probability 30% each, as demonstrated in the results section.

## 5.2 Decision trees for beta contact features

Beta contact features are less suited to the lasso regression approach than torsion angle features, because independence assumptions are not as valid. For instance, contact $(i, j, \text{parallel})$ and contact

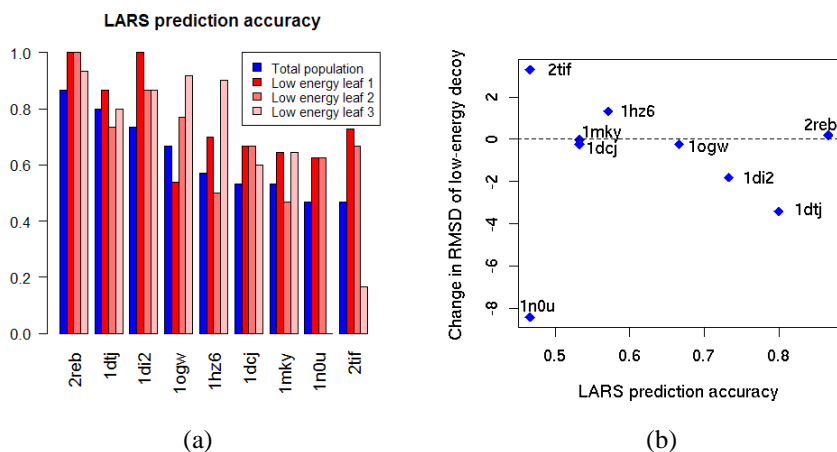

(a)  (b)

Figure 4: (a) LARS prediction accuracy when fitted to total decoy population and to the three decision-tree leaves with lowest 10th percentile energies, ordered here by average rmsd. (b) Relation of prediction accuracy to resampling improvement in LARS-only runs.

$(i + 1, j + 1, \text{parallel})$ are redundant and will usually co-occur, whereas contact $(i, j, \text{parallel})$ and contact $(i - 1, j + 1, \text{parallel})$ are mutually exclusive and will never co-occur. For beta contact features, we employ a decision tree approach to divide the decoy population into non-overlapping clusters each defined by the presence of several beta contacts. Lasso regression is then employed in each cluster separately to determine likely native torsion features.

We use decision trees of depth three. At each node, a beta contact feature is selected to use as a split point and a child node is created for each of the three possible values X, P1, and P2. Our strategy is to choose split points which most reduce entropy in the features. The beta contact feature is therefore chosen whose mutual information with the other beta contact features is maximized, as approximated by the sum of the marginal mutual informations with each other feature.

Since some clusters are sampled more heavily than others, the lowest energy within a cluster is not a fair measure of its quality, even though, in principle, we care only about the lowest achievable energy. Instead, we use the $10^{\text{th}}$ percentile energy to evaluate clusters. Its advantage as a statistic is that its expectation is not dependent on sample size, but it often gives a reasonably tight upper bound on achievable energy. Our resampling strategy, given a decision tree, is to sample evenly from each of the top three leaves as ranked by $10^{\text{th}}$ percentile energy. Within the subpopulation of decoys defined by each leaf, we select torsion features using LARS.

In our benchmark set, the top three low-energy leaves of the decision tree were generally closer to the native than the population at large. Perhaps as a result, LARS generally achieved greater prediction accuracy when restricted to their associated subpopulations, as seen in Figure 3.b. Leaves are sorted by average rmsd, so "low energy leaf 1," the "best" leaf, consists of decoys which are closest, on average, to the native conformation. The best leaf consisted of only native contacts for all proteins except 1n0u and 1ogw, but in both these cases it contained structures generally lower in rmsd than the population at large and resampling achieved improvements over plain Rosetta sampling. In general, LARS performed better on the leaves that were closer to the native structure, although there were a few notable exceptions. Ideally, we would concentrate our sampling entirely on the best leaf, but since we cannot generally identify which one it is, we have to hedge our bets. Including more leaves in the resampling round increases the chances of resampling a native leaf but dilutes sampling of the best leaf in the pool. This tradeoff is characteristic of resampling methods.

## 6 Results

We tested two Rosetta resampling schemes over a set of 9 alpha/beta proteins of between 59 and 81 residues. In the first scheme (referred to henceforth as "LARS-only"), 15 LARS-predicted torsion features were constrained at 30% frequency. In the second (referred to henceforth as "decision-

| | RMSD of low-energy decoys | | | | Lowest RMSD of 25 low-energy decoys | | | |
| | Decision tree | | LARS only | | Decision tree | | LARS only | |
| | Control | Resamp | Control | Resamp | Control | Resamp | Control | Resamp |
|---|---|---|---|---|---|---|---|---|
| 1di2 | 2.35 | 2.14 | 2.76 | 0.97 | 1.78 | 1.34 | 1.82 | 0.73 |
| 1dtj | 3.20 | 1.53 | 5.28 | 1.88 | 1.46 | 1.53 | 1.95 | 1.59 |
| 1dcj | 2.35 | 3.31 | 2.34 | 2.11 | 2.19 | 1.86 | 1.71 | 1.88 |
| 1ogw | 5.22 | 3.99 | 3.03 | 2.80 | 3.12 | 2.6 | 2.08 | 2.48 |
| 2reb | 1.15 | 1.17 | 1.07 | 1.27 | 0.89 | 0.93 | 0.83 | 0.86 |
| 2tif | 5.68 | 4.57 | 3.57 | 6.85 | 3.32 | 3.27 | 3.27 | 2.61 |
| 1n0u | 11.89 | 11.60 | 11.93 | 3.54 | 9.78 | 3.19 | 3.54 | 2.84 |
| 1hz6A | 2.52 | 1.06 | 3.36 | 4.68 | 2.38 | 1.06 | 1.97 | 1.19 |
| 1mkyA | 10.39 | 8.21 | 4.60 | 4.58 | 3.43 | 3.25 | 3.33 | 4.23 |
| Mean difference | | -0.8 | | -1.03 | | -1.04 | | -0.23 |
| Median difference | | -1.11 | | -0.23 | | -0.33 | | -0.36 |
| Number improved | | 7/9 | | 6/9 | | 7/9 | | 5/9 |

tree"), three subpopulations were defined for each protein using a decision tree, and within each subpopulation 15 LARS-predicted torsion features were constrained at frequencies heuristically determined on the basis of several meta-level "features of features," including the rate of the feature's occurrence in the first round of Rosetta sampling and the magnitude of the regression weight for the feature. Each resampling scheme was compared against a control population generated at the same time. Exactly the same number of models were generated for the control and resampled populations. The control and resampled populations for the LARS-only scheme consist of about 200,000 decoys each. The populations for the decision-tree scheme consist of about 30,000 decoys each, due to limitations in available compute time. The difference in quality between the two control populations is partially explained by the different numbers of samples in each, and partially by changes in Rosetta in the time between the generation of the two datasets.

Our two primary measures of success for a resampling run are both based on root-mean-square distance to the native structure. Root-mean-square distance (rmsd) is a standard measure of discrepancy between two structures. It is defined as the square root of the mean of the squared distances between pairs of corresponding backbone atoms in the two structures, under the alignment that minimizes this quantity. Our first measure of success is the rmsd between the native structure and the lowest scoring model. This measures Rosetta's performance if forced to make a single prediction. Our second measure of success is lowest rmsd among the twenty-five top-scoring models. This is a smoother measure of the quality of the lowest scoring Rosetta models, and gives some indication of the prediction quality if more sophisticated minima-selection methods are used than Rosetta energy ranking. Structures at 1Å from the native have atomic-level resolution—this is the goal. Structures at between 2Å and 4Å generally have several important structural details incorrect. In proteins the size of those in our benchmark, structures more than 5Å from the native are poor predictions.

Both resampling schemes achieved some success. The performance measures are shown in table 6. The decision-tree scheme performed more consistently and achieved larger improvements on average; it improved the low-energy rmsd in 7 of the 9 benchmark proteins, with a significant median improvement of 1.11Å. Particularly exciting are the atomic-resolution prediction for 1hz6 and the nearly atomic-resolution prediction for 1dtj. In both these cases, plain Rosetta sampling performed considerably worse. The LARS-only scheme was successful as well, providing improved lowest-energy predictions on 6 of the 9 benchmark proteins with a median improvement of 0.23Å. The LARS-only low-energy prediction for 1di2 is atomic-resolution at 0.97Å away from the native structure, as compared to 2.97Å for the control run. In general, improvements correlated with LARS accuracy (Figure 4.b). The two notable exceptions were 2reb, for which plain Rosetta search performs so well that constraints only hurt sampling, and 1n0u, for which plain Rosetta search concentrates almost entirely on a cluster with incorrect topology at 10Å. Certain LARS-selected features, when enforced, switch sampling over to a cluster at around 3Å. Even when incorrect features are enforced within this cluster, sampling is much improved.

The cases in which the decision-tree scheme did not yield improved low-energy predictions are interesting in their own right. In the case of 1dcj, resampling does yield lower rmsd structures—the top 25 low rms prediction is superior, and the minimum rmsd from the set is 1.35, nearly atomic

resolution, as compared to 1.95 for the control run—but the Rosetta energy function does not pick them out. This suggests that better decoy-selection techniques would improve our algorithms. In the case of 2reb, the unbiased rounds of Rosetta sampling were so successful that they would have been difficult to improve on. This emphasizes the point that resampling cannot hurt us too much. If a plain Rosetta sampling round of $n$ decoys is followed by a resampling round of $n$ decoys, then no matter how poor the resampled decoys are, sampling efficiency is decreased by at most a factor of 2 (since we could have generated $n$ plain Rosetta samples in the same time). The danger is that resampling may overconverge to broad, false energy wells, achieving lower energies in the resampling round even though rmsd is higher. This appears to occur with 2tif, in which the LARS-only low-energy prediction has significantly lower energy than the control prediction despite being much farther from the native. Once more, better decoy-selection techniques might help.

# 7  Discussion and Conclusions

Our results demonstrate that our resampling techniques improve structure prediction on a majority of the proteins in our benchmark set. Our first resampling method significantly improves Rosetta predictions in 3 of the 9 test cases, and marginally improves two or three more. Our second resampling method expands the set of proteins on which we achieve improvements, including an additional atomic-level prediction. It is important to note that significant improvements over Rosetta on *any* proteins are hard to achieve; if our methods achieved one or two significantly improved predictions, we would count them a success. Rosetta is the state of the art in protein structure prediction, and it has undergone years of incremental advances and optimizations. Surpassing its performance is very difficult. Furthermore, it doesn't hurt Rosetta too badly if a resampling scheme performs worse than unbiased sampling on some proteins, since models from the unbiased sampling round that precedes the resampling round can be used as predictions as well.

There are a number of avenues of future work to pursue. We have designed a number of other structural features, including per-residue secondary structure features, burial features, and side-chain rotamer features, and we hope to incorporate these into our methods. The primary barrier is that each new feature requires a method for constraining it during Rosetta search. We also plan to further investigate the possibility of detecting which LARS predictions are correct using "features of features," and to apply these methods to discrimate between decision tree leaves as well. It is possible that, with more sampling, the decision tree runs would yield atomic-resolution predictions. However, computational costs for Rosetta are high; each Rosetta model takes approximately fifteen minutes of CPU time to compute on a 1GHz CPU, and each of the 36 data sets represented here consists of on the order of $100,000$ models.

The success of our feature selection techniques suggests that the high dimensionality and multiple minima that make high resolution protein structure prediction difficult to solve using traditional methods provide an excellent application for modern machine learning methods. The intersection between the two fields is just beginning, and we are excited to see further developments.

# References

[1] G. E. P. Box and K. B. Wilson. On the experimental attainment of optimum conditions (with discussion). *Journal of the Royal Statistical Society Series B*, 13(1):1–45, 1951.

[2] Justin Boyan and Andrew W. Moore. Learning evaluation functions to improve optimization by local search. *The Journal of Machine Learning Research*, 1:77–112, 2001.

[3] Phil Bradley, Lars Malmstrom, Bin Qian, Jack Schonbrun, Dylan Chivian, David E. Kim, Jens Meiler, Kira M. Misura, and David Baker. Free modeling with Rosetta in CASP6. *Proteins*, 61(S7):128–134, 2005.

[4] Bradley Efron, Trevor Hastie, Iain Johnstone, and Robert Tibshirani. Least angle regression. *Annals of Statistics (with discussion)*, 32(2):407–499, 2004.

[5] Wolfgang Kabsch and Chris Sander. Dictionary of protein secondary structure: pattern recognition of hydrogen-bonded and geometrical features. *Biopolymers*, 22(12):2577–2637, 1983.

[6] Robert Tibshirani. Regression shrinkage and selection via the lasso. *Journal of the Royal Statistical Society Series B*, 58(1):267–288, 1996.

